# A LOW-POWER CMOS CIRCUIT WHICH EMULATES TEMPORAL ELECTRICAL PROPERTIES OF NEURONS

Jack L. Meador and Clint S. Cole
Electrical and Computer Engineering Dept.
Washington State University
Pullman WA. 99164-2752

## ABSTRACT

This paper describes a CMOS artificial neuron. The circuit is directly derived from the voltage-gated channel model of neural membrane, has low power dissipation, and small layout geometry. The principal motivations behind this work include a desire for high performance, more accurate neuron emulation, and the need for higher density in practical neural network implementations.

## INTRODUCTION

Popular neuron models are based upon some statistical measure of known natural behavior. Whether that measure is expressed in terms of average firing rate or a firing probability, the instantaneous neuron activation is only represented in an abstract sense. Artificial electronic neurons derived from these models represent this excitation level as a binary code or a continuous voltage at the output of a summing amplifier. While such models have been shown to perform well for many applications, and form an integral part of much current work, they only partially emulate the manner in which natural neural networks operate. They ignore, for example, differences in relative arrival times of neighboring action potentials -- an important characteristic known to exist in natural auditory and visual networks {Sejnowski, 1986}. They are also less adaptable to fine-grained, neuron-centered learning, like the post-tetanic facilitation observed in natural neurons. We are investigating the implementation and application of neuron circuits which better approximate natural neuron function.

## BACKGROUND

The major temporal artifacts associated with natural neuron function include the spacio-temporal integration of synaptic activity, the generation of an action potential (AP), and the post-AP hyperpolarization (refractory) period (Figure 1). Integration, manifested as a gradual membrane depolarization, occurs when the neuron accumulates sodium ions which migrate through pores in its cellular membrane. The rate of ion migration is related to the level of presynaptic AP bombardment, and is also known to be a non-linear function of transmembrane potential. Efferent AP generation occurs when the voltage-sensitive membrane of the axosomal hillock reaches some threshold potential whereupon a rapid increase in sodium permeability leads to

complete depolarization. Immediately thereafter, sodium pores "close" simultaneously with increased potassium permeability, thereby repolarizing the membrane toward its resting potential. The high potassium permeability during AP generation leads to the transient post-AP hyperpolarization state known as the refractory period.

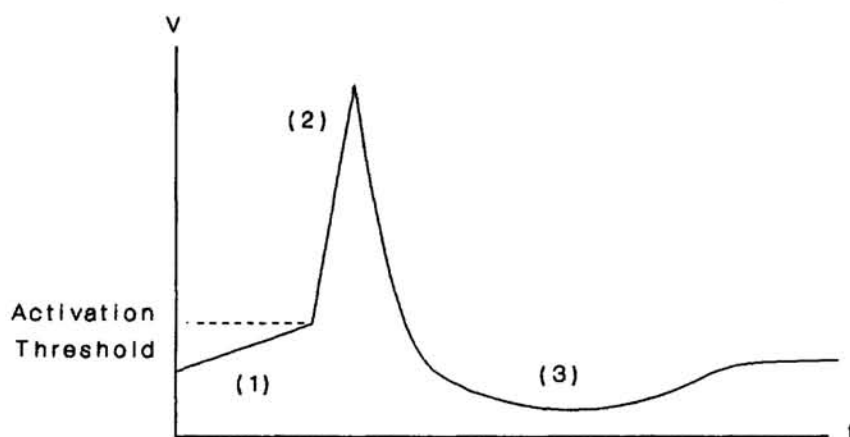

**Figure 1.** Temporal artifacts associated with neuron function. (1) gradual depolarization, (2) AP generation, (3) refractory period.

Several analytic and electronic neural models have been proposed which embody these characteristics at varying levels of detail. These neuromimes have been used to good advantage in studying neuron behavior. However, with the advent of artificial neural networks (ANN) for computing, emphasis has switched from modeling neurons for physiologic studies to developing practical neural network implementations. As the desire for high performance ANNs grows, models amenable to hardware implementation become more attractive.

The general idea behind electronic neuromimes is not new. Beginning in 1937 with work by Harmon {Harmon, 1937}, electronic circuits have been used to model and study neuronal behavior. In the late 1960's, Lewis {Lewis, 1968} developed a circuit which simulated the Hodgkin-Huxley model for a single neuron, followed by MacGregor's circuit {MacGregor, 1973} in the early 1970's which modelled a group of 50 neurons. With the availability of VLSI in the 1980's, electronic neural implementations have largely moved to the realm of integrated circuits. Two different strategies have been documented: analog implementations employing operational amplifiers {Graf, et al, 1987,1988; Sivilotti, et al, 1986; Raffel, 1988; Schwartz, et al, 1988}; and digital implementations such as systolic arrays {Kung, 1988}.

More recently, impulse neural implementations are receiving increased attention. Like other models, these neuromimes generate outputs based on some non-linear function of the weighted net inputs. However, interneuron communication is realized through impulse streams rather than continuous voltages or binary numbers {Murray, 1988; N. El-Leithy, 1987}. Impulse networks communicate neuron activation as variable pulse repetition rates. The impulse neuron circuits which shall be discussed offer both small geometry and low power dissipation as well as a closer approximation to natural neuron function.

# A CMOS IMPULSE NEURON

An impulse neuron circuit developed for use in CMOS neural networks is shown in Figure 2. In this circuit, membrane ion current is modeled by charge flowing to and from $C_a$. Potassium and sodium influx is represented by current flow from $V_{dd}$ to the capacitor, and ion efflux by flow from the capacitor to ground. The Field Effect-Transistors (FETs) connected between $V_{dd}$, $V_{ss}$, and the capacitor emulate voltage- and chemically-gated ion channels found in natural neural membrane. In the Figure, FET 1 corresponds to the post-synaptic chemically-gated ion channels associated with one synapse. FETs 2, 3, and 4 emulate the voltage-gated channels distributed throughout a neuron membrane. The following equations summarize circuit operation:

$$C_a \, dV_a/dt = \beta_1 E(V_s, V_a) + \beta_{23} F(V_a) - \beta_4 G(V_a) \tag{1}$$

$$E(V_s, V_a) = (V_s - V_a - V_{tn})(V_{dd} - V_a) - (V_{dd} - V_a)^2/2 \tag{2}$$

$$F(V_a) = \begin{cases} (V_{dd} - V_{tp})(V_a - V_{dd}) - (V_a - V_{dd})^2/2 & \text{if } g(t) = 0 \\ 0 & \text{otherwise} \end{cases} \tag{3}$$

$$G(V_a) = \begin{cases} (V_{dd} - V_{tn})V_a - V_a^2/2 & \text{if } h(t - \delta) = 0 \\ 0 & \text{otherwise} \end{cases} \tag{4}$$

$$g(t) = h(t)(1 - h(t - \delta)) \tag{5}$$

$$h(t) = \begin{cases} 0 & \text{if } V_a(t) > V_{th}; \\ & V_{tl} < V_a(t) < V_{th} \text{ and } h(V_a(t - \varepsilon)) = 0 \\ 1 & \text{if } V_a(t) < V_{tl}; \\ & V_{tl} < V_a(t) < V_{th} \text{ and } h(V_a(t - \varepsilon)) = 1 \end{cases} \tag{6}$$

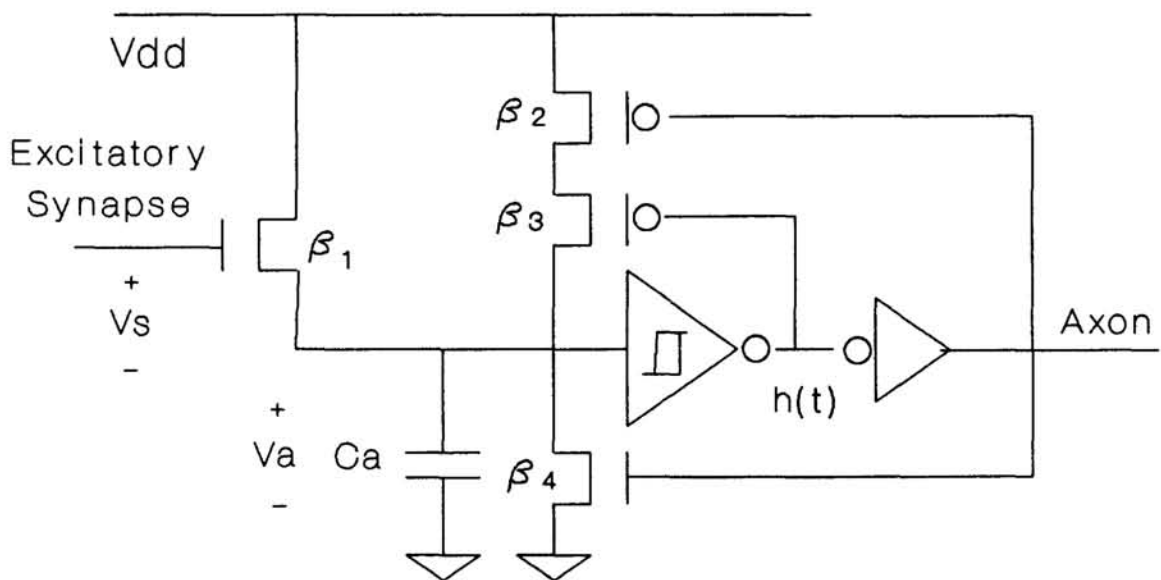

**Figure 2.** A CMOS impulse neuron with one excitatory synapse-FET.

Equation (1) expresses how changes in $V_a$ (which emulates instantaneous neuron excitation) depend upon the sum of three current components controlled by these FETs. $E$, $F$, and $G$ in equations (2) through (4) express FET drain-source currents as functions terminal voltages. Equations (3) and (5) rely upon the assumption that FET 2 and FET 3 are implemented as a single dual-gate device where the transconductance $\beta_{23} = \beta_2\beta_3/(\beta_2+\beta_3)$. Non-saturated FET operation is assumed for these equations even though the FETs will momentarily pass through saturation at the onset of conduction in the actual circuit.

The Schmitt trigger circuit establishes a nonlinear positive feedback path responsible for action potential initiation. The upper threshold of the trigger ($V_{th}$) emulates the natural neuron activation threshold while the lower threshold ($V_{tl}$) emulates the maximum hyperpolarization voltage. Equation (6) expresses the hysterisis present in the Schmitt trigger transfer characteristic. When $V_s$ reaches the upper Schmitt threshold, FET 3 turns on, creating a current path from $V_{dd}$ to $C_s$, and emulating the upswing of a natural action potential spike. A moment later, FET 2 turns off, starting the action potential downswing. Simultaneously, FET 4 turns on, begining the absolute refractory period where $C_s$ is discharged toward the maximum hyperpolarization potential. When that potential is reached, the Schmitt trigger turns off FET 4 and the impulse firing cycle is complete.

The capacitor terminal voltage $V_a$ emulates all gross temporal artifacts associated with membrane potential, including spacio-temporal integration, the action potential spike, and a refractory period. The instantaneous net excitation to the neuron is represented by the total current flowing into the summing node on the floating plate of the capacitor. Charge packets are transferred from $V_{dd}$ to the capacitor by the excitatory synapse FET. Excitatory packet magnitude is dependent upon the transconductance $\beta_1$. Inhibitory synapses (not shown) operate similarly, but instead reduce capacitor voltage by drawing charge to $V_{ss}$. A buffered action potential signal useful for driving many synapse FETs is available at the *axon* output.

The membrane potential components ($E$, $F$, and $G$) of the circuit equations describe nonlinear relationships between post-synaptic excitation ($E$), membrane potential ($F$ and $G$), and membrane ion currents. The functional forms of these components are equivalent to those found between terminal voltages and currents in non-saturated FETs. It is notable that natural voltage-gated channels do not necessarily follow the same current-voltage relationship of a FET. Even though more accurate models and emulations of natural membrane conductance exist, it seems unlikely at this time that they would help further improve neural network implementation. There is little doubt that more complex circuitry would be required to better approximate the true non-linear relationship found in the biochemistry of natural neural membrane. That need conflicts directly with the goal of high-density integration.

## IMPULSE NEURAL NETWORKS

Organizing a collection of neuron circuits into a useful network configuration requires some weight specification method. Weight values can be either directly specified by the designer or learned by the network. A method particularly suited for use with the fixed FET-synapses of the foregoing circuit is to first learn weights using an "off-line"

simulation, then translate the numerical results to physical FET transconductances. To do this, the activation function of an impulse neuron is derived and used in a modified back-propagation learning procedure.

## IMPULSE NEURON ACTIVATION FUNCTION

Learning algorithms typically require some expression of the neuron activation function. Neuron activation can be expressed as a numerical value, a binary pattern, or a circuit voltage. In an impulse neuron, activation is expressed in terms of firing rate. The more frequently an impulse neuron circuit fires, the greater its activation. Impulse neuron activation is a nonlinear function of the excitation imparted through its synapse connections. An analytical expression of this nonlinear function can be derived using a rectangular approximation of neuron impulse waveforms.

It is first necessary to define a unit-impulse as one impulse conducted by a synapse FET having some pre-determined reference transconductance ($\beta_{ref}$). In Figure 3, $T_0$ represents an invariant activation impulse width which is assumed to be identical for all neurons. $T_1$ represents the variable time period required for the neuron to accumulate the equivalent of $K$ unit-impulses input excitation prior to firing. It can be assumed that net input comes from a single excitatory synapse with no other excitation. It shall also be assumed that impulses arrive at a constant rate, so

$$T_1 = K/w_{ij}R_i \tag{7}$$

where $R_i$ is the firing rate of the source neuron and $w_{ij}$ is the weight of the synapse connecting neuron $i$ and neuron $j$.

The firing rate of the receiving neuron will be $R_j = 1/(T_0 + T_1)$. Substituting for $T_1$ this becomes:

$$R_j = 1/(T_0 + K/w_{ij}R_i) = w_{ij}R_i/(w_{ij}R_iT_0 + K) \tag{8}$$

Figure 3 compares this function with the logistic activation function. The impulse activation function approaches zero at the rate of $1/K$ when $R_i$ approaches zero. The function also approaches an asymptote of $R_j = 1/T_0$ as $R_i$ increases without bound. Any non-synaptic source which causes current flow from $V_{dd}$ to $C_a$ will shift the curve to the left, and reflect a spontaneous firing rate at zero input excitation. A similar current source to $V_{ss}$ will shift the function to the right, reflecting a positive firing-onset threshold. Circuit-level simulations show a clear correspondence to these analytical results. This functional form is also evident in activation curves experimentally observed with natural neurons {Guyton, 1986}. Various natural neurons are known to exhibit both spontaneous firing and firing-onset thresholds as well.

The impulse activation function constant, K, is determined by several factors including $\beta_{ref}$, $C_a$, and $T_0$. Assuming that $T_0 \ll T_1$, no leakage current exists, and that a FET conducting in its non-saturated region can be approximated by a resistor, the following expression for K is obtained:

$$K \approx R_{chan}C_a \ln((Vdd - V_{tl})/(Vdd - V_{th}))/T_0 \tag{9}$$

where

$$R_{chan} \approx 1/\beta_{ref}(Vdd - V_{tn}),$$

$C_a$ is the summing capacitance, $V_{tl}$ $V_{th}$ are the low and high threshold voltages of the Schmitt trigger, and $V_{tn}$ is the gate threshold voltage for an excitatory FET-synapse. A more accurate K value can be obtained by using the non-saturated FET current equation and solving a nonlinear differential equation.

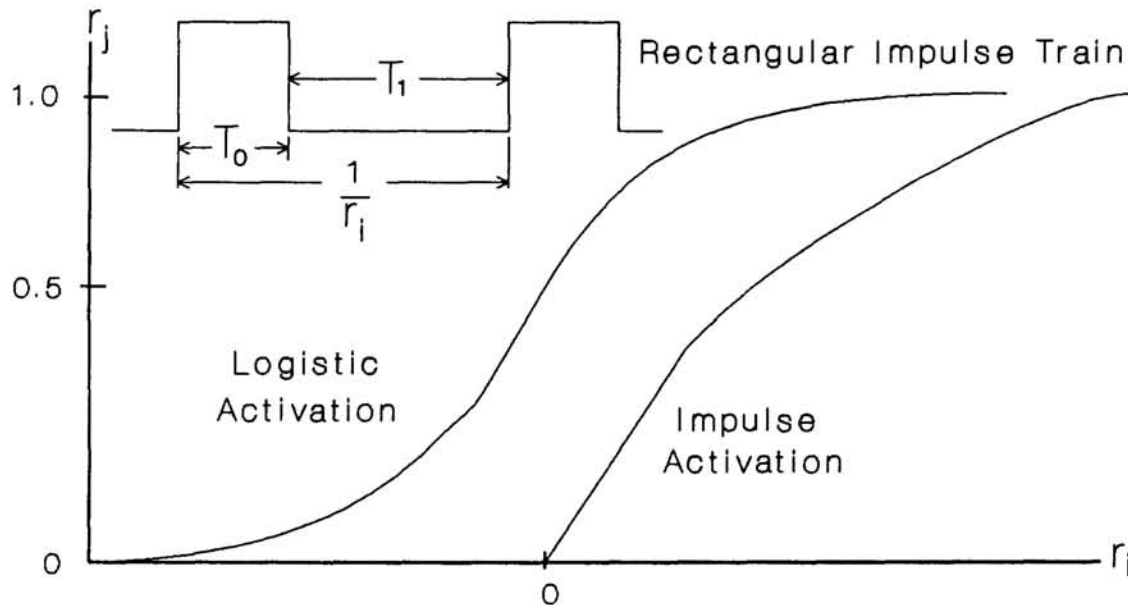

**Figure 3.** Rectangular impulse train approximation for impulse activation function derivation. Unlike the logistic function which asymptotically approaches zero, impulse activation is equal to zero over a range of net excitation.

## BACK-PROPAGATION IN IMPULSE NETWORKS

A back-propagation algorithm has been used to learn connection weights for impulse neural networks. At this time, weight values are non-adaptive (they are fixed at circuit fabrication) because they are implemented as invariant FET transconductances. Adaptive synapses compatible with impulse neuron circuits are in the early stages of development, but are not available at this time. Much can be learned about these networks using non-adaptive prototypes, however. As a result, weight learning is performed offline as part of the network design process. The back-propagation procedure used to learn weights for impulse networks differs from the generalized delta rule {Rumelhart, 1986} in two ways.

The first difference is the use of the impulse activation function instead of the logistic function. Any activation nonlinearity is a viable candidate for use with the generalized delta rule as long as it is differentiable. This is where difficulties mount with the impulse activation function. First of all, it is not differentiable at zero. What seems to be more important, however, is that its first derivative equals zero over a range of

inputs. Examination of the generalized delta rule (which performs gradient-descent) reveals that when the first derivative of neuron activation becomes zero, connections associated with that neuron will cease to adapt. Once this happens, the procedure will most probably never arrive at a problem solution.

To work around this problem, a second deviation from the generalized delta rule was implemented. This involves a departure from using the true first derivative when the impulse activation becomes zero. A small constant can be used to guarantee that learning continues even though the associated neuron activation is zero:

$$Act = 1/(T_0 + K/Net) \tag{10}$$

$$Act' = \begin{cases} (1/(T_0 + K/Net)' & \text{if } Net > 0 \\ \varepsilon & \text{otherwise} \end{cases} \tag{11}$$

The use of these equations yields a back-propagation algorithm for impulse networks which does not perform true gradient descent, yet which so far has been observed to learn solutions to logic problems such as XOR and the 4-2-4 encoder. Investigation of other offline learning algorithms for impulse networks continues. Currently, this algorithm fulfills the immediate need for an offline procedure which can be used in the design of multi-layer impulse neural networks.

## IMPLEMENTATION

Two requirements for high density integration are low power dissipation and small circuit geometry. CMOS impulse neurons use switching circuits having no continuous power dissipation. A conventional op-amp circuit must draw constant current to achieve linear bias. An op-amp also requires larger circuit geometries for gain accuracy over typical fabrication process variations. Such is not the case for nonlinear switching circuits. As a result, these neurons and others like them are expected to help improve analog neural network integration density.

An impulse neuron circuit has been designed which eliminates FETs 2 and 3 of Figure 2 in exchange for reduced layout area. In this circuit, $V_a$ no longer exhibits an activation potential spike. This spike seems irrelevant given the buffered impulse available at the *axon* output. The modified neuron circuit occupies 200 X 25 lambda chip area. A fixed FET-synapse occupies a 16 by 18 lambda rectangle. With these dimensions a full-interconnect layout containing 40 neurons and 1600 fixed connections will fit on a MOSIS 2-micron tiny chip. XOR and 4-2-4 networks of these circuits are being developed for 2-micron CMOS.

## CONCLUSION

The motivation of this work is to improve neural network implementation technology by designing CMOS circuits derived from the temporal characteristics of natural neurons. The results obtained thus far include:

Two CMOS circuits which closely correspond to the voltage-gated-channel model of natural neural membrane.

Simulations which show that these impulse neurons emulate gross artifacts of natural neuron function.

Initial work on a back-propagation algorithm which learns logic solutions using the impulse neuron activation function.

The development of prototype impulse network I.Cs.

Future goals involve extending this investigation to plastic synapse and neuron circuits, alternate algorithms for both offline and online learning, and practical implementations.

**References**

H. P. Graf W. Hubbard L. D. Jackel P. G. N. deVegvar. A CMOS associative Memory Chip. *IEEE ICNN Con. Proc.*, pp. 461-468, (1987).

H. P. Graf L. D. Jackel W. E. Hubbard. VLSI Implementation of a Neural Network Model. *IEEE Computer*, pp. 41-49, (1988).

A.C. Guyton. Chapt. 10. Organization of the Nervous System: Basic Functions of Synapses. *Textbook of Physiology*, p.136. (1986)

N. El-Liethy, R.W. Newcomb, M. Zaghlou. A Basic MOS Neural-Type Junction A Perspective on Neural-Type Microsystems. *IEEE ICNN Con. Proc.*, pp. 469-477, (1987).

E. R. Lewis. Using Electronic Circuits to Model Simple Neuroelectric Interactions. *Proc. IEEE 56*, pp. 931-949, (1968).

R. J. MacGregor R. M. Oliver. A General-Purpose Electronic Model for Arbitrary Configurations of Neurons. *J. Theor. Biol. 38*, pp. 527-538 (1973).

S. Y. Kung. and J. N. Hwang. Parallel Achitectures for Artificial Neural Nets. *IEEE ICNN Con. Proc.*, pp. II-165 to II-172, (1988).

J. Mann R. Lippman B. Berger J. Raffel. A Self-Organizing Neural Net Chip. *IEEE Cust. Integr. Ckts. Conf.*, pp. 10.3.1-10.3.5 (1988).

A. F. Murray A. V. W. Smith. Asynchronous VLSI Neural Networks Using Pulse-Stream Arithmetic. *IEEE Jnl. of Sol. St. Phys. 23*, pp. 688-697, (1988).

J. I. Raffel. Electronic Implementation of Neuromorphic Systems. *IEEE Cust. Integr. Ckts. Conf.*, pp. 10.1.1-10.1.7, (1988).

D. Rumelhart, G.E. Hinton, and R.J. Williams. Learning Internal Representations by Error Propagation. *Parallel Distributed Processing, Vol 1: Foundations*, pp. 318-364, (1986).

O. H. Schmitt. Mechanical Solution of the Equations of Nerve Impulse Propagation. *Am. J. Physiol. 119*, pp. 399-400, (1937).

D. B. Schwartz R. E. Howard. A Programmable Analog Neural Network Chip. *IEEE Cust. Integr. Ckts. Conf.*, pp. 10.2.1-1.2.4, (1988).

T.J. Sejnowski. Open Questions About Computation in Cerebral Cortex. *Parallel Distributed Processing Vol. 2:Psychological and Biological Models*, pp. 378-385, (1986).

M. A. Sivilotti  M. R. Emerling  C. A. Mead. VLSI Architectures for Implementation of Neural Networks. *Am. Ins. of Phys.*, 408-413, (1986).

.